# Outlier Detection with One-class Kernel Fisher Discriminants

**Volker Roth**
ETH Zurich, Institute of Computational Science
Hirschengraben 84, CH-8092 Zurich
*vroth@inf.ethz.ch*

## Abstract

The problem of detecting "atypical objects" or "outliers" is one of the classical topics in (robust) statistics. Recently, it has been proposed to address this problem by means of one-class SVM classifiers. The main conceptual shortcoming of most one-class approaches, however, is that in a strict sense they are unable to *detect* outliers, since the expected fraction of outliers has to be specified in advance. The method presented in this paper overcomes this problem by relating kernelized one-class classification to Gaussian density estimation in the induced feature space. Having established this relation, it is possible to identify "atypical objects" by quantifying their deviations from the Gaussian model. For RBF kernels it is shown that the Gaussian model is "rich enough" in the sense that it asymptotically provides an unbiased estimator for the true density. In order to overcome the inherent model selection problem, a cross-validated likelihood criterion for selecting all free model parameters is applied.

## 1 Introduction

A one-class-classifier attempts to find a separating boundary between a data set and the rest of the feature space. A natural application of such a classifier is estimating a contour line of the underlying data density for a certain quantile value. Such contour lines may be used to separate "typical" objects from "atypical" ones. Objects that look "sufficiently atypical" are often considered to be *outliers*, for which one rejects the hypothesis that they come from the same distribution as the majority of the objects. Thus, a useful application scenario would be to find a boundary which separates the jointly distributed objects from the outliers. Finding such a boundary defines a classification problem in which, however, usually only sufficiently many labeled samples from one class are available. Usually no *labeled samples* from the outlier class are available at all, and it is even unknown if there are *any* outliers present.

It is interesting to notice that the approach of directly estimating a boundary, as opposed to first estimating the whole density, follows one of the main ideas in learning theory which states that one should avoid solving a too hard intermediate problem. While this line of reasoning seems to be appealing from a theoretical point of view, it leads to a severe problem in practical applications: when it comes to *detecting* outliers, the restriction to estimating only a boundary makes it impossible to derive a formal characterization of outliers without prior assumptions on the *expected fraction of outliers* or even on their distribution. In practice, however, any such prior assumptions can hardly be justified. The fundamental

problem of the one-class approach lies in the fact that outlier detection is a (partially) unsupervised task which has been "squeezed" into a classification framework. The missing part of information has been shifted to prior assumptions which can probably only be justified, if the solution of the original problem was known in advance.

This paper aims at overcoming this problem by linking kernel-based one-class classifiers to Gaussian density estimation in the induced feature space. Objects which have an "unexpected" high Mahalanobis distance to the sample mean are considered as "atypical objects" or outliers. A particular Mahalanobis distance is considered to be unexpected, if it is very unlikely to observe an object that far away from the mean vector in a random sample of a certain size. We will formalize this concept in section 3 by way of fitting linear models in quantile-quantile plots. The main technical ingredient of our method is the *one-class kernel Fisher discriminant* classifier (OC-KFD), for which the relation to Gaussian density estimation is shown. From the classification side, the OC-KFD-based model inherits the simple complexity control mechanism by using regularization techniques. The explicit relation to Gaussian density estimation, on the other hand, makes it possible to formalize the notion of atypical objects by observing deviations from the Gaussian model. It is clear that these deviations will heavily depend on the chosen model parameters. In order to derive an *objective* characterization of atypical objects it is, thus, necessary to select a *suitable* model in advance. This model-selection problem is overcome by using a likelihood-based cross-validation framework for inferring the free parameters.

## 2 Gaussian density estimation and one-class LDA

Let $X$ denote the $n \times d$ data matrix which contains the $n$ input vectors $\boldsymbol{x}_i \in \mathbb{R}^d$ as rows. It has been proposed to estimate a one-class decision boundary by separating the dataset from the origin [12], which effectively coincides with replicating all $\boldsymbol{x}_i$ with opposite sign and separating $X$ and $-X$. Typically, a $\nu$-SVM classifier with RBF kernel function is used. The parameter $\nu$ bounds the expected number of outliers and must be selected a priori. The method proposed here follows the same idea of separating the data from their negatively replicated counterparts. Instead of a SVM, however, a Kernel Fisher Discriminant (KFD) classifier is used [7, 10]. The latter has the advantage that is is closely related to Gaussian density estimation in the induced feature space. By making this relation explicit, outliers can be identified without specifying the expected fraction of outliers in advance. We start with a linear discriminant analysis (LDA) model, and then kernels will be introduced.

Let $X' = (X, -X)^\top$ denote the augmented $(2n \times d)$ data matrix which also contains the negative samples $-\boldsymbol{x}_i$. Without loss of generality we assume that the sample mean $\boldsymbol{\mu}_+ \equiv \sum_i \boldsymbol{x}_i > 0$, so that the sample means of the positive data and the negative data differ: $\boldsymbol{\mu}_+ \neq \boldsymbol{\mu}_-$. Let us now assume that our data are realizations of a normally distributed random variable in $d$ dimensions: $\mathcal{X} \sim N_d(\boldsymbol{\mu}, \Sigma)$. Denoting by $X^c$ the centered data matrix, the estimator for $\Sigma$ takes the form $\hat{\Sigma} \equiv W = (1/n) X^{c\top} X^c$.

The LDA solution $\boldsymbol{\beta}_*$ maximizes the between-class scatter $\boldsymbol{\beta}_*^\top B \boldsymbol{\beta}_*$ with $B = \boldsymbol{\mu}_+ \boldsymbol{\mu}_+^\top + \boldsymbol{\mu}_- \boldsymbol{\mu}_-^\top$ under the constraint on the within-class scatter $\boldsymbol{\beta}_*^\top W \boldsymbol{\beta}_* = 1$. Note that in our special case with $X' = (X, -X)^\top$ the usual pooled within-class matrix $W$ simply reduces to the above defined $W = (1/n) X^{c\top} X^c$. Denoting by $\boldsymbol{y}' = (2, \dots, 2, -2, \dots, -2)^\top$ a $2n$-indicator vector for class membership in class "+" or "−", it is well-known (see e.g. [1]) that the LDA solution (up to a scaling factor) can be found by minimizing a least-squares functional: $\hat{\boldsymbol{\beta}} = \arg\min_{\boldsymbol{\beta}} \|\boldsymbol{y}' - X'\boldsymbol{\beta}\|^2$. In [3] a slightly more general form of the problem is described where the above functional is minimized under a constrained on $\boldsymbol{\beta}$, which in the simplest case amounts to adding a term $\gamma \boldsymbol{\beta}^\top \boldsymbol{\beta}$ to the functional. Such a *ridge regression* model assumes a penalized total covariance of the form $T = (1/(2n)) \cdot X'^\top X' + \gamma I = (1/n) \cdot X^\top X + \gamma I$. Defining an $n$-vector of ones $\boldsymbol{y} = (1, \dots, 1)^\top$, the solution vector $\hat{\boldsymbol{\beta}}$

reads

$$\hat{\boldsymbol{\beta}} = (X'^\top X' + \gamma I)^{-1} X'^\top \boldsymbol{y}' = (X^\top X + \gamma I)^{-1} X^\top \boldsymbol{y}. \tag{1}$$

According to [3], an appropriate scaling factor is defied in terms of the quantity $s^2 = (1/n) \cdot \boldsymbol{y}^\top \hat{\boldsymbol{y}} = (1/n) \cdot \boldsymbol{y}^\top X \hat{\boldsymbol{\beta}}$, which leads us to the correctly scaled LDA vector $\boldsymbol{\beta}_* = s^{-1}(1-s^2)^{-1/2}\hat{\boldsymbol{\beta}}$ that fulfills the normalization condition $\boldsymbol{\beta}_*^\top W \boldsymbol{\beta}_* = 1$.

One further derives from [3] that the mean vector of $X$, projected onto the 1-dimensional LDA-subspace has the coordinate value $m_+ = s(1-s^2)^{-1/2}$, and that the Mahalanobis distance from a vector $\boldsymbol{x}$ to the sample mean $\boldsymbol{\mu}_+$ is the sum of the squared Euclidean distance in the projected space and an orthogonal distance term:

$$D(\boldsymbol{x}, \boldsymbol{\mu}_+) = (\boldsymbol{\beta}_*^\top \boldsymbol{x} - m_+)^2 + D_\perp \text{ with } D_\perp = -(1-s^2)(\boldsymbol{\beta}_*^\top \boldsymbol{x})^2 + \boldsymbol{x}^\top T^{-1} \boldsymbol{x}. \tag{2}$$

Note that it is the term $D_\perp$ which makes the density estimation model essentially different from OC-classification: while the latter considers only distances in the direction of the projection vector $\boldsymbol{\beta}$, the true density model additionally takes into account the distances in the orthogonal subspace.

Since the assumption $\mathcal{X} \sim N_d(\boldsymbol{\mu}, \Sigma)$ is very restrictive, we propose to relax it by assuming that we have found a suitable transformation of our input data $\boldsymbol{\phi} : \mathbb{R}^d \mapsto \mathbb{R}^p$, $\boldsymbol{x} \mapsto \boldsymbol{\phi}(\boldsymbol{x})$, such that the *transformed* data are Gaussian in $p$ dimensions. If the transformation is carried out implicitly by introducing a Mercer kernel $k(\boldsymbol{x}_i, \boldsymbol{x}_j)$, we arrive at an equivalent problem in terms of the kernel matrix $K = \Phi\Phi^\top$ and the expansion coefficients $\boldsymbol{\alpha}$:

$$\hat{\boldsymbol{\alpha}} = (K + \gamma I)^{-1} \boldsymbol{y}. \tag{3}$$

From [11] it follows that the mapped vectors can be represented in $\mathbb{R}^n$ as $\boldsymbol{\phi}(\boldsymbol{x}) = K^{-1/2} \boldsymbol{k}(\boldsymbol{x})$, where $\boldsymbol{k}(\boldsymbol{x})$ denotes the kernel vector $\boldsymbol{k}(\boldsymbol{x}) = (k(\boldsymbol{x}, \boldsymbol{x}_1), \dots, k(\boldsymbol{x}, \boldsymbol{x}_n))^\top$. Finally we derive the following form of the Mahalanobis distances which again consist of the Euclidean distance in the classification subspace plus an orthogonal term:

$$D(\boldsymbol{x}, \boldsymbol{\mu}_+) = (\boldsymbol{\alpha}_*^\top \boldsymbol{k}(\boldsymbol{x}) - m_+)^2 - (1-s^2)(\boldsymbol{\alpha}_*^\top \boldsymbol{k}(\boldsymbol{x}))^2 + n\Omega(\boldsymbol{x}), \tag{4}$$

where $\Omega(\boldsymbol{x}) = \boldsymbol{\phi}^\top(\boldsymbol{x})(\Phi^\top\Phi + \gamma I)^{-1}\boldsymbol{\phi}(\boldsymbol{x}) = \boldsymbol{k}^\top(\boldsymbol{x})(K + \gamma I)^{-1}K^{-1}\boldsymbol{k}(\boldsymbol{x})$, $m_+ = s(1-s^2)^{-1/2}$, $s^2 = (1/n) \cdot \boldsymbol{y}^\top \hat{\boldsymbol{y}} = (1/n) \cdot \boldsymbol{y}^\top K \hat{\boldsymbol{\alpha}}$, and $\boldsymbol{\alpha}_* = s^{-1}(1-s^2)^{-1/2}\hat{\boldsymbol{\alpha}}$.

Equation (4) establishes the desired link between OC-KFD and Gaussian density estimation, since for our outlier detection mechanism only Mahalanobis distances are needed. While it seems to be rather complicated to estimate a density by the above procedure, the main benefit over directly estimating the mean and the covariance lies in the inherent complexity regulation properties of ridge regression. Such a complexity control mechanism is of particular importance in highly nonlinear kernel models. Moreover, for ridge-regression models it is possible to analytically calculate the *effective degrees of freedom*, a quantity that will be of particular interest when it comes to detecting outliers.

## 3 Detecting outliers

Let us assume that the model is completely specified, i.e. both the kernel function $k(\cdot, \cdot)$ and the regularization parameter $\gamma$ are fixed. The central lemma that helps us to detect outliers can be found in most statistical textbooks:

**Lemma 1.** *Let $\mathcal{X}$ be a Gaussian random variable $\mathcal{X} \sim N_d(\boldsymbol{\mu}, \Sigma)$. Then $\Delta \equiv (\mathcal{X} - \boldsymbol{\mu})^\top \Sigma^{-1} (\mathcal{X} - \boldsymbol{\mu})$ follows a chi-square ($\chi^2$) distribution on $d$ degrees of freedom.*

For the penalized regression models, it might be more appropriate to use the *effective* degrees of freedom $df$ instead of $d$ in the above lemma. In the case of one-class LDA with ridge penalties we can easily estimate it as $df = trace(X(X^\top X + \gamma I)^{-1} X^\top)$, [8], which

for a kernel model translates into $df = trace(K(K + \gamma I)^{-1})$. The intuitive interpretation of the quantity $df$ is the following: denoting by $V$ the matrix of eigenvectors of $K$ and by $\{\lambda_i\}_{i=1}^n$ the corresponding eigenvalues, the fitted values $\hat{y}$ read

$$\hat{y} = V \text{diag}\left\{\delta_i = \lambda_i/(\lambda_i + \gamma)\right\} V^\top y. \tag{5}$$

It follows that compared to the unpenalized case, where all eigenvectors $v_i$ are constantly weighted by 1, the contribution of the $i$-th eigenvector $v_i$ is down-weighted by a factor $\delta_i/1 = \delta_i$. If the ordered eigenvalues decrease rapidly, however, the values $\delta_i$ are either close to zero or close to one, and $df$ determines the number of terms that are "essentially different" from zero. The same is true for the orthogonal distance term in eq. (4): note that

$$\Omega(\boldsymbol{x}) = \boldsymbol{k}^\top(\boldsymbol{x})(K+\gamma I)^{-1} K^{-1} \boldsymbol{k}(\boldsymbol{x}) = \boldsymbol{k}^\top V \text{diag}\left\{\delta_i' = ((\lambda_i + \gamma)\lambda_i)^{-1}\right\} V^\top \boldsymbol{k}(\boldsymbol{x}). \tag{6}$$

Compared to the unpenalized case (the contribution of $v_i$ is weighted by $\lambda_i^{-2}$), the contribution of $v_i$ is down-weighted by the same factor $\delta_i'/\lambda_i^{-2} = \delta_i$.

From lemma 1 we conclude that if the data are well described by a Gaussian model in the kernel feature space, the observed Mahalanobis distances should look like a sample from a $\chi^2$-distribution with $df$ degrees of freedom. A graphical way to test this hypothesis is to plot the observed quantiles against the theoretical $\chi^2$ quantiles, which in the ideal case gives a straight line. Such a *quantile-quantile plot* is constructed as follows: Let $\Delta_{(i)}$ denote the observed Mahalanobis distances ordered from lowest to highest, and $p_i$ the *cumulative proportion* before each $\Delta_{(i)}$ given by $p_i = (i - 1/2)/n$. Let further $z_i = F^{-1} p_i$ denote the theoretical quantile at position $p_i$, where $F$ is the cumulative $\chi^2$-distribution function. The quantile-quantile plot is then obtained by plotting $\Delta_{(i)}$ against $z_i$. Deviations from linearity can be formalized by fitting a linear model on the observed quantiles and calculating confidence intervals around the fit. Observations falling outside the confidence interval are then treated as outliers. A potential problem of this approach is that the outliers themselves heavily influence the quantile-quantile fit. In order to overcome this problem, the use of *robust* fitting procedures has been proposed in the literature, see e.g. [4]. In the experiments below we use an M-estimator with Huber loss function. For estimating confidence intervals around the fit we use the standard formula (see [2, 5])

$$\sigma(\Delta_{(i)}) = b \cdot (\chi^2(z_i))^{-1} \sqrt{(p_i(1 - p_i))/n}, \tag{7}$$

which can be intuitively understood as the product of the slope $b$ and the standard error of the quantiles. A $100(1 - \varepsilon)\%$ envelope around the fit is then defined as $\Delta_{(i)} \pm z_{\varepsilon/2}\sigma(\Delta_{(i)})$ where $z_{\varepsilon/2}$ is the $1 - (1 - \varepsilon)/2$ quantile of the standard normal distribution.

The choice of the confidence level $\varepsilon$ is somewhat arbitrary, and from a conceptual viewpoint one might even argue that the problem of specifying one free parameter (i.e. the expected fraction of outliers) has simply been transferred into the problem of specifying another one. In practice, however, selecting $\varepsilon$ is a much more intuitive procedure than guessing the fraction of outliers. Whereas the latter requires problem-specific prior knowledge which is hardly available in practice, the former depends only on the variance of a linear model fit. Thus, $\varepsilon$ can be specified in a *problem independent* way.

## 4    Model selection

In our model the data are first mapped into some feature space, in which then a Gaussian model is fitted. Mahalanobis distances to the mean of this Gaussian are computed by evaluating (4). The feature space mapping is implicitly defined by the kernel function, for which we assume that it is parametrized by a kernel parameter $\sigma$. For selecting all free parameters in (4), we are, thus, left with the problem of selecting $\boldsymbol{\theta} = (\sigma, \gamma)^\top$.

The idea is now to select $\boldsymbol{\theta}$ by maximizing the *cross-validated* likelihood. From a theoretical viewpoint, the cross-validated (CV) likelihood framework is appealing, since in [13]

the CV likelihood selector has been shown to asymptotically perform as well as the optimal *benchmark selector* which characterizes the best possible model (in terms of Kullback-Leibler divergence to the true distribution) contained in the parametric family.

For kernels that map into a space with dimension $p > n$, however, two problems arise: (i) the subspace spanned by the mapped samples varies with different sample sizes; (ii) not the whole feature space is accessible for vectors in the input space. As a consequence, it is difficult to find a "proper" normalization of the Gaussian density in the induced feature space. We propose to avoid this problem by considering the likelihood in the *input space* rather than in the feature space, i.e. we are looking for a properly normalized density model $p(\boldsymbol{x}|\cdot)$ in $\mathbb{R}^d$ such that $p(\boldsymbol{x}|\cdot)$ has the same contour lines as the Gaussian model in the feature space: $p(\boldsymbol{x}_i|\cdot) = p(\boldsymbol{x}_i|\cdot) \Leftrightarrow p(\boldsymbol{\phi}(\boldsymbol{x}_i)|\cdot) = p(\boldsymbol{\phi}(\boldsymbol{x}_j)|\cdot)$. Denoting by $X_n = \{\boldsymbol{x}_i\}_{i=1}^n$ a sample from $p(\boldsymbol{x})$ from which the kernel matrix $K$ is built, a natural input space model is

$$p_n(\boldsymbol{x}|X_n, \boldsymbol{\theta}) = Z^{-1} \exp\{-\tfrac{1}{2}D(\boldsymbol{x}; X_n, \boldsymbol{\theta})\}, \text{ with } Z = \int_{\mathbb{R}^d} p_n(\boldsymbol{x}|X_n, \boldsymbol{\theta})\,d\boldsymbol{x}, \quad (8)$$

where $D(\boldsymbol{x}; X_n, \boldsymbol{\theta})$ denotes the (parametrized) Mahalanobis distances (4) of a Gaussian model in the feature space. Note that this density model in the input space has the same form as our Gaussian model in the feature space, except for the different normalization constant $Z$. Computing this constant $Z$ requires us to solve a normalization integral over the whole $d$-dimensional input space. Since in general this integral is not analytically tractable for nonlinear kernel models, we propose to approximate $Z$ by a Monte Carlo sampling method. In our experiments, for instance, the VEGAS algorithm [6], which implements a mixed importance-stratified sampling approach, showed to be a reasonable method for up to 10 input dimensions.

By using the CV likelihood framework we are guaranteed to (asymptotically) perform as well as the best model in the parametrized family. Thus, the question arises whether the family of densities defined by a Gaussian model in a kernel-induced feature space is "rich enough" such that no systematic errors occur. For RBF kernels, the following lemma provides a positive answer to this question.

**Lemma 2.** *Let* $k(\boldsymbol{x}_i, \boldsymbol{x}_j) = \exp(-\|\boldsymbol{x}_i - \boldsymbol{x}_j\|^2/\sigma)$. *As* $\sigma \to 0$, $p_n(\boldsymbol{x}|X_n, \boldsymbol{\theta})$ *converges to a Parzen window with vanishing kernel width:* $p_n(\boldsymbol{x}|X_n, \boldsymbol{\theta}) \to \frac{1}{n}\sum_{i=1}^n \delta(\boldsymbol{x} - \boldsymbol{x}_i)$.

A formal proof is omitted due to space limitations. The basic ingredients of the proof are: (i) In the limit $\sigma \to 0$ the expansion coefficients approach $\hat{\alpha} \to 1/(1 + \gamma)\mathbf{1}$. Thus, $\hat{\boldsymbol{y}} = K\hat{\alpha} \to 1/(1 + \gamma)\mathbf{1}$ and $s^2 \to 1/(1 + \gamma)$. (ii) $D(\boldsymbol{x}; \sigma, \gamma) \to C(\boldsymbol{x}) < \infty$, if $\boldsymbol{x} \in \{\boldsymbol{x}_i\}_{i=1}^n$, and $D(\boldsymbol{x}; \sigma, \gamma) \to \infty$, else. Finally $p_n(\boldsymbol{x}|X_n, \sigma, \gamma) \to \frac{1}{n}\sum_{i=1}^n \delta(\boldsymbol{x} - \boldsymbol{x}_i)$.

Note that in the limit $\sigma \to 0$ a Parzen window becomes an unbiased estimator for any continuous density, which provides an asymptotic justification for our approach: the cross-validated likelihood framework guarantees us to convergence to a model that performs as well as the best model in our model class as $n \to \infty$. The latter, however, is "rich enough" in the sense that it contains models which in the limit $\sigma \to 0$ converge to an unbiased estimator for every continuous $p(\boldsymbol{x})$. Since contour lines of $p_n(\boldsymbol{x})$ are contour lines of a Gaussian model in the feature space, the Mahalanobis distances are expected to follow a $\chi^2$ distribution, and atypical objects can be detected by observing the distribution of the empirical Mahalanobis distances as described in the last section.

It remains to show that describing the data as a Gaussian in a kernel-induced feature space is a statistically sound model. This is actually the case, since there exist decay rates for the kernel width $\sigma$ such that $n$ grows at a higher rate as the effective degrees of freedom $df$:

**Lemma 3.** *Let* $k(\boldsymbol{x}_i, \boldsymbol{x}_j) = \exp(-\|\boldsymbol{x}_i - \boldsymbol{x}_j\|^2/\sigma)$ *and* $p_n(\boldsymbol{x}|X_n, \sigma, \gamma)$ *defined by (8). If* $\sigma \leq 1$ *decays like* $O(n^{-1/2})$, *and for fixed* $\gamma \leq 1$, *the ratio* $df/n \to 0$ *as* $n \to \infty$.

A formal proof is omitted due to space limitations. The basic ingredients of the proof are: (i) the eigenvalues $\lambda_i'$ of $(1/n)K$ converge to $\bar{\lambda}_i$ as $n \to \infty$, (ii) the eigenvalue spectrum of

a Gaussian RBF kernel decays at an exponential-quadratic rate: $\bar{\lambda}_i \propto \exp(-\sigma i^2)$, (iii) for $n$ sufficiently large, it holds that $\sum_{i=1}^{n} 1/[1 + (\gamma/n) \exp(n^{-1/2} \sigma i^2)] \leq n^{1/2} \sigma^{-1} \log(n/\gamma)$ (proof by induction, using the fact that $\ln(n+1) - \ln(n) \geq 1/(n^2 + n)$ which follows from a Taylor expansion of the logarithm) $\Rightarrow df(n)/n \to 0$.

## 5 Experiments

The performance of the proposed method is demonstrated for an outlier detection task in the field of face recognition. The Olivetti face database (see http://www.uk.research.att.com/facedatabase.html) contains ten different images of each of 40 distinct subjects, taken under different lighting conditions and at different facial expressions and facial details (glasses / no glasses). None of the subjects, however, wears sunglasses. All the images are taken against a homogeneous background with the subjects in an upright, frontal position. In this experiment we additionally corrupted the dataset by including two images in which we have artificially changed normal glasses to "sunglasses" as can be seen in figure 1. The goal is to demonstrate that the proposed method is able to identify these two atypical images without any problem-dependent prior assumptions.

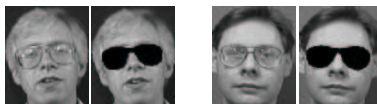

Figure 1: Original and corrupted images with in-painted "sunglasses".

Each of the 402 images is characterized by a 10-dimensional vector which contains the projections onto the leading 10 eigenfaces (eigenfaces are simply the eigenvectors of the images treated as pixel-wise vectorial objects). These vectors are feed into a RBF kernel of the form $k(\boldsymbol{x}_i, \boldsymbol{x}_j) = \exp(-\|\boldsymbol{x}_i - \boldsymbol{x}_j\|^2/\sigma)$. In a first step, the free model parameters $(\sigma, \gamma)$ are selected by maximizing the cross-validated likelihood. A simple 2-fold cross validation scheme is used: the dataset is randomly split into a training set and a test set of equal size, the model is build from the training set (including the numerical solution of the normalization integral), and finally the likelihood is evaluated on the test set. This procedure is repeated for different values of $(\sigma, \gamma)$. In order to simplify the selection process we kept $\gamma = 10^{-4}$ fixed and varied only $\sigma$. Both the test likelihood and the corresponding model complexity measured in terms of the effective degrees of freedom ($df$) are plotted in figure 2. One can clearly identify both an overfitting and an underfitting regime, separated by a broad plateau of models with similarly high likelihood. The $df$-curve, however, shows a similar plateau, indicating that all these models have comparable complexity. This observation suggests that the results should be rather insensitive to variations of $\sigma$ over values contained in this plateau. This suggestion is indeed confirmed by the results in figure 2, where we compared the quantile-quantile plot for the maximum likelihood parameter value with that of a slightly suboptimal model. Both quantile plots look very similar, and in both cases two objects clearly fall outside a 99% envelope around the linear fit. Outside the plateau (no figure due to space limitations) the number of objects considered as outlies drastically increases in overfitting regime ($\sigma$ too small), or decreases to zero in the underfitting regime ($\sigma$ too large).

In figure 3 again the quantile plot for the most likely model is depicted. This time, however, both objects identified as outliers are related to the corresponding original images, which in fact are the artificially corrupted ones. In addition, the uncorrupted images are localized in the plot, indicating that they look rather typical.

**Some implementation details.** Presumably the easiest way of implementing the model is to carry out an eigenvalue decomposition of $K$. Both the the effective degrees of freedom $df = \sum_i \lambda_i/(\lambda_i + \gamma)$ and the Mahalanobis distances in eq. (4) can then by derived easily

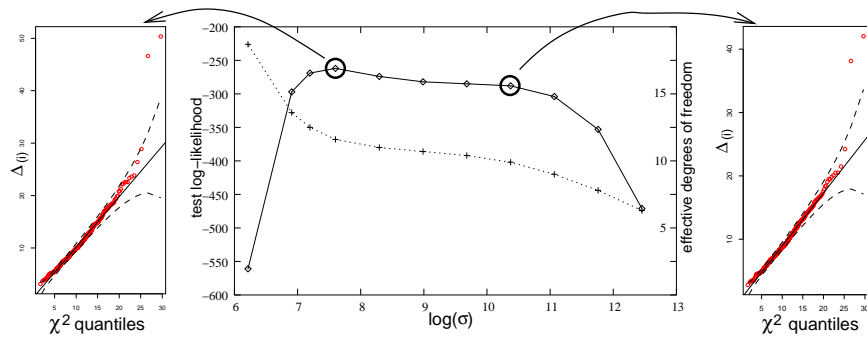

Figure 2: Middle panel: Selecting the kernel width $\sigma$ by cross-validated likelihood (solid line). The dotted line shows the corresponding effective degrees of freedom $(df)$. Left + right panels: quantile plot for optimal model (left) and slightly suboptimal model (right).

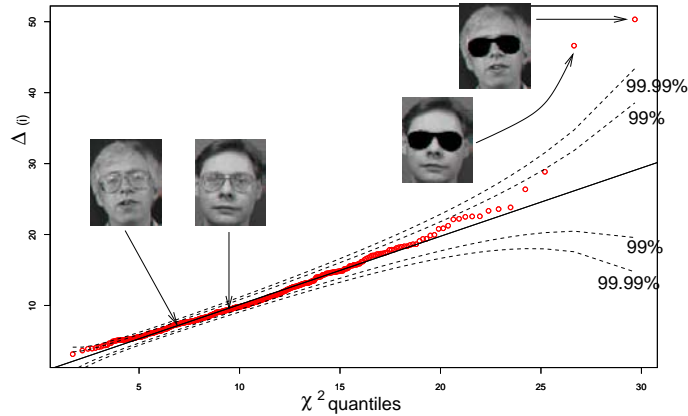

Figure 3: Quantile plot with linear fit (solid) and envelopes (99% and 99.99 %, dashed).

from this decomposition (see (5) and (6)). Efficient on-line variants can be implemented by using standard update formulas for matrix inversion by partitioning. For an implementation of the VEGAS algorithm see [9]. The R package "car" provides a comfortable implementation of quantile-quantile plots and robust line fitting (see also http://www.R-project.org).

## 6 Conclusion

Detecting outliers by way of one-class classifiers aims at finding a boundary that separates "typical" objects in a data sample from the "atypical" ones. Standard approaches of this kind suffer from the problem that they require prior knowledge about the expected fraction of outliers. For the purpose of outlier *detection*, however, the availability of such prior information seems to be an unrealistic (or even contradictory) assumption. The method proposed in this paper overcomes this shortcoming by using a one-class KFD classifier which is directly related to Gaussian density estimation in the induced feature space. The model benefits from both the built-in classification method and the explicit parametric density model: from the former it inherits the simple complexity regulation mechanism based on only two tuning parameters. Moreover, within the classification framework it is possible to quantify the model complexity in terms of the effective degrees of freedom $df$. The Gaussian density model, on the other hand, makes it possible to derive a formal description of atypical objects by way of hypothesis testing: Mahalanobis distances are expected to follow a $\chi^2$-distribution in $df$ dimensions, and deviations from this distribution can be

quantified by confidence intervals around a fitted line in a quantile-quantile plot. Since the density model is parametrized by both the kernel function and the regularization constant, it is necessary to select these free parameters *before* the outlier detection phase. This parameter selection is achieved by observing the cross-validated likelihood for different parameter values, and choosing those parameters which maximize this quantity. The theoretical motivation for this selection process follows from [13] where it has been shown that the cross-validation selector asymptotically performs as well as the so called *benchmark selector* which selects the best model contained in the parametrized family of models. Moreover, for RBF kernels it is shown in lemma 2 that the corresponding model family is "rich enough" in the sense that it contains an unbiased estimator for the true density (as long as it is continuous) in the limit of vanishing kernel width. Lemma 3 shows that there exist decay rates for the kernel width such that the ratio of effective degrees of freedom and sample size approaches zero.

The experiment on detecting persons wearing sunglasses within a collection of rather heterogeneous face images effectively demonstrates that the proposed method is able to detect atypical objects without prior assumptions on the expected number of outliers. In particular, it demonstrates that the whole processing pipeline consisting of model selection by cross-validated likelihood, fitting linear quantile-quantile models and detecting outliers by considering confidence intervals around the fit works very well in practical applications with reasonably small input dimensions. For input dimensions $\gg 10$ the numerical solution of the normalization integral becomes rather time consuming when using the VEGAS algorithm. Evaluating the usefulness of more sophisticated sampling models like Markov-Chain Monte-Carlo methods for this particular task will be subject of future work.

**Acknowledgments.** The author would like to thank Tilman Lange, Mikio Braun and Joachim M. Buhmann for helpful discussions and suggestions.

## References

[1] R. Duda, P. Hart, and D. Stork. *Pattern Classification*. Wiley & Sons, 2001.

[2] J. Fox. *Applied Regression, Linear Models, and Related Methods*. Sage, 1997.

[3] T. Hastie, A. Buja, and R. Tibshirani. Penalized discriminant analysis. *Annals of Statistics*, 23:73–102, 1995.

[4] P.J. Huber. *Robust Statistics*. Wiley, 1981.

[5] M. Kendall and A. Stuart. *The Advanced Theory of Statistics*, volume 1. McMillan, 1977.

[6] G.P. Lepage. Vegas: An adaptive multidimensional integration program. Technical Report CLNS-80/447, Cornell University, 1980.

[7] S. Mika, G. Rätsch, J. Weston, B. Schölkopf, and K.-R. Müller. Fisher discriminant analysis with kernels. In Y.-H. Hu, J. Larsen, E. Wilson, and S. Douglas, editors, *Neural Networks for Signal Processing IX*, pages 41–48. IEEE, 1999.

[8] J. Moody. The effective number of parameters: An analysis of generalisation and regularisation in nonlinear learning systems. In J. Moody, S. Hanson, and R. Lippmann, editors, *NIPS 4*, 1992.

[9] W.H. Press, S.A. Teukolsky, W.T. Vetterling, and B.P. Flannery. *Numerical Recipies in C*. Cambridge University Press, 1992.

[10] V. Roth and V. Steinhage. Nonlinear discriminant analysis using kernel functions. In S.A. Solla, T.K. Leen, and K.-R. Müller, editors, *NIPS 12*, pages 568–574. MIT Press, 2000.

[11] B. Schölkopf, S. Mika, C. Burges, P. Knirsch, K.-R. Müller, G. Rätsch, and A. Smola. Input space vs. feature space in kernel-based methods. *IEEE Trans. Neural Networks*, 10(5), 1999.

[12] B. Schölkopf, R.C. Williamson, A. Smola, and J. Shawe-Taylor. SV estimation of a distribution's support. In S. Solla, T. Leen, and K.-R. Müller, editors, *NIPS 12*, pages 582–588. 2000.

[13] M.J. van der Laan, S. Dudoit, and S. Keles. Asymptotic optimality of likelihood-based cross-validation. *Statistical Applications in Genetics and Molecular Biology*, 3(1), 2004.
